# Online Metric Learning and Fast Similarity Search

**Prateek Jain, Brian Kulis, Inderjit S. Dhillon, and Kristen Grauman**
Department of Computer Sciences
University of Texas at Austin
Austin, TX 78712
{pjain,kulis,inderjit,grauman}@cs.utexas.edu

## Abstract

Metric learning algorithms can provide useful distance functions for a variety
of domains, and recent work has shown good accuracy for problems where the
learner can access all distance constraints at once. However, in many real appli-
cations, constraints are only available incrementally, thus necessitating methods
that can perform online updates to the learned metric. Existing online algorithms
offer bounds on worst-case performance, but typically do not perform well in
practice as compared to their offline counterparts. We present a new online metric
learning algorithm that updates a learned Mahalanobis metric based on LogDet
regularization and gradient descent. We prove theoretical worst-case performance
bounds, and empirically compare the proposed method against existing online
metric learning algorithms. To further boost the practicality of our approach, we
develop an online locality-sensitive hashing scheme which leads to efficient up-
dates to data structures used for fast approximate similarity search. We demon-
strate our algorithm on multiple datasets and show that it outperforms relevant
baselines.

## 1   Introduction

A number of recent techniques address the problem of metric learning, in which a distance function
between data objects is learned based on given (or inferred) similarity constraints between exam-
ples [4, 7, 11, 16, 5, 15]. Such algorithms have been applied to a variety of real-world learning
tasks, ranging from object recognition and human body pose estimation [5, 9], to digit recogni-
tion [7], and software support [4] applications. Most successful results have relied on having access
to all constraints at the onset of the metric learning. However, in many real applications, the desired
distance function may need to change gradually over time as additional information or constraints
are received. For instance, in image search applications on the internet, online click-through data
that is continually collected may impact the desired distance function. To address this need, recent
work on *online* metric learning algorithms attempts to handle constraints that are received one at a
time [13, 4]. Unfortunately, current methods suffer from a number of drawbacks, including speed,
bound quality, and empirical performance.

Further complicating this scenario is the fact that fast retrieval methods must be in place on top
of the learned metrics for many applications dealing with large-scale databases. For example, in
image search applications, relevant images within very large collections must be quickly returned
to the user, and constraints and user queries may often be intermingled across time. Thus a good
online metric learner must also be able to support fast similarity search routines. This is problematic
since existing methods (e.g., locality-sensitive hashing [6, 1] or kd-trees) assume a static distance
function, and are expensive to update when the underlying distance function changes.

The goal of this work is to make metric learning practical for real-world learning tasks in which both constraints and queries must be handled efficiently in an online manner. To that end, we first develop an online metric learning algorithm that uses LogDet regularization and exact gradient descent. The new algorithm is inspired by the metric learning algorithm studied in [4]; however, while the loss bounds for the latter method are dependent on the input data, our loss bounds are independent of the sequence of constraints given to the algorithm. Furthermore, unlike the Pseudo-metric Online Learning Algorithm (POLA) [13], another recent online technique, our algorithm requires no eigenvector computation, making it considerably faster in practice. We further show how our algorithm can be integrated with large-scale approximate similarity search. We devise a method to incrementally update locality-sensitive hash keys during the updates of the metric learner, making it possible to perform accurate sub-linear time nearest neighbor searches over the data in an online manner.

We compare our algorithm to related existing methods using a variety of standard data sets. We show that our method outperforms existing approaches, and even performs comparably to several offline metric learning algorithms. To evaluate our approach for indexing a large-scale database, we include experiments with a set of 300,000 image patches; our online algorithm effectively learns to compare patches, and our hashing construction allows accurate fast retrieval for online queries.

## 1.1 Related Work

A number of recent techniques consider the metric learning problem [16, 7, 11, 4, 5]. Most work deals with learning Mahalanobis distances in an offline manner, which often leads to expensive optimization algorithms. The POLA algorithm [13], on the other hand, is an approach for online learning of Mahalanobis metrics that optimizes a large-margin objective and has provable regret bounds, although eigenvector computation is required at each iteration to enforce positive definiteness, which can be slow in practice. The information-theoretic metric learning method of [4] includes an online variant that avoids eigenvector decomposition. However, because of the particular form of the online update, positive-definiteness still must be carefully enforced, which impacts bound quality *and* empirical performance, making it undesirable for both theoretical and practical purposes. In contrast, our proposed algorithm has strong bounds, requires no extra work for enforcing positive definiteness, and can be implemented efficiently. There are a number of existing online algorithms for other machine learning problems outside of metric learning, e.g. [10, 2, 12].

Fast search methods are becoming increasingly necessary for machine learning tasks that must cope with large databases. Locality-sensitive hashing [6] is an effective technique that performs approximate nearest neighbor searches in time that is sub-linear in the size of the database. Most existing work has considered hash functions for $L_p$ norms [3], inner product similarity [1], and other standard distances. While recent work has shown how to generate hash functions for (offline) learned Mahalanobis metrics [9], we are not aware of any existing technique that allows incremental updates to locality-sensitive hash keys for online database maintenance, as we propose in this work.

## 2 Online Metric Learning

In this section we introduce our model for online metric learning, develop an efficient algorithm to implement it, and prove regret bounds.

### 2.1 Formulation and Algorithm

As in several existing metric learning methods, we restrict ourselves to learning a *Mahalanobis distance function* over our input data, which is a distance function parameterized by a $d \times d$ positive definite matrix $A$. Given $d$-dimensional vectors $\boldsymbol{u}$ and $\boldsymbol{v}$, the squared Mahalanobis distance between them is defined as

$$d_A(\boldsymbol{u}, \boldsymbol{v}) = (\boldsymbol{u} - \boldsymbol{v})^T A(\boldsymbol{u} - \boldsymbol{v}).$$

Positive definiteness of $A$ assures that the distance function will return positive distances. We may equivalently view such distance functions as applying a linear transformation to the input data and computing the squared Euclidean distance in the transformed space; this may be seen by factorizing the matrix $A = G^T G$, and distributing $G$ into the $(\boldsymbol{u} - \boldsymbol{v})$ terms.

In general, one learns a Mahalanobis distance by learning the appropriate positive definite matrix $A$ based on constraints over the distance function. These constraints are typically distance or similarity constraints that arise from supervised information—for example, the distance between two points in the same class should be "small". In contrast to offline approaches, which assume all constraints

are provided up front, online algorithms assume that constraints are received one at a time. That is, we assume that at time step $t$, there exists a current distance function parameterized by $A_t$. A constraint is received, encoded by the triple $(\boldsymbol{u}_t, \boldsymbol{v}_t, y_t)$, where $y_t$ is the target distance between $\boldsymbol{u}_t$ and $\boldsymbol{v}_t$ (we restrict ourselves to distance constraints, though other constraints are possible). Using $A_t$, we first *predict* the distance $\hat{y}_t = d_{A_t}(\boldsymbol{u}_t, \boldsymbol{v}_t)$ using our current distance function, and incur a loss $\ell(\hat{y}_t, y_t)$. Then we *update* our matrix from $A_t$ to $A_{t+1}$. The goal is to minimize the sum of the losses over all time steps, i.e. $L_A = \sum_t \ell(\hat{y}_t, y_t)$. One common choice is the squared loss: $\ell(\hat{y}_t, y_t) = \frac{1}{2}(\hat{y}_t - y_t)^2$. We also consider a variant of the model where the input is a quadruple $(\boldsymbol{u}_t, \boldsymbol{v}_t, y_t, b_t)$, where $b_t = 1$ if we require that the distance between $\boldsymbol{u}_t$ and $\boldsymbol{v}_t$ be less than or equal to $y_t$, and $b_t = -1$ if we require that the distance between $\boldsymbol{u}_t$ and $\boldsymbol{v}_t$ be greater than or equal to $y_t$. In that case, the corresponding loss function is $\ell(\hat{y}_t, y_t, b_t) = \max(0, \frac{1}{2}b_t(\hat{y}_t - y_t))^2$.

A typical approach [10, 4, 13] for the above given online learning problem is to solve for $A_{t+1}$ by minimizing a regularized loss at each step:

$$A_{t+1} = \operatorname*{argmin}_{A \succ 0} D(A, A_t) + \eta \ell(d_A(\boldsymbol{u}_t, \boldsymbol{v}_t), y_t), \tag{2.1}$$

where $D(A, A_t)$ is a regularization function and $\eta_t > 0$ is the regularization parameter. As in [4], we use the *LogDet* divergence $D_{\ell d}(A, A_t)$ as the regularization function. It is defined over positive definite matrices and is given by $D_{\ell d}(A, A_t) = \operatorname{tr}(AA_t^{-1}) - \log \det(AA_t^{-1}) - d$. This divergence has previously been shown to be useful in the context of metric learning [4]. It has a number of desirable properties for metric learning, including scale-invariance, automatic enforcement of positive definiteness, and a maximum-likelihood interpretation.

Existing approaches solve for $A_{t+1}$ by approximating the gradient of the loss function, i.e. $\ell'(d_A(\boldsymbol{u}_t, \boldsymbol{v}_t), y_t)$ is approximated by $\ell'(d_{A_t}(\boldsymbol{u}_t, \boldsymbol{v}_t), y_t)$ [10, 13, 4]. While for some regularization functions (e.g. Frobenius divergence, von-Neumann divergence) such a scheme works out well, for LogDet regularization it can lead to non-definite matrices for which the regularization function is not even defined. This results in a scheme that has to adapt the regularization parameter in order to maintain positive definiteness [4].

In contrast, our algorithm proceeds by *exactly* solving for the updated parameters $A_{t+1}$ that minimize (2.1). Since we use the exact gradient, our analysis will become more involved; however, the resulting algorithm will have several advantages over existing methods for online metric learning. Using straightforward algebra and the Sherman-Morrison inverse formula, we can show that the resulting solution to the minimization of (2.1) is:

$$A_{t+1} = A_t - \frac{\eta(\bar{y} - y_t) A_t \boldsymbol{z}_t \boldsymbol{z}_t^T A_t}{1 + \eta(\bar{y} - y_t) \boldsymbol{z}_t^T A_t \boldsymbol{z}_t}, \tag{2.2}$$

where $\boldsymbol{z}_t = \boldsymbol{u}_t - \boldsymbol{v}_t$ and $\bar{y} = d_{A_{t+1}}(\boldsymbol{u}_t, \boldsymbol{v}_t) = \boldsymbol{z}_t^T A_{t+1} \boldsymbol{z}_t$. The detailed derivation will appear in a longer version. It is not immediately clear that this update can be applied, since $\bar{y}$ is a function of $A_{t+1}$. However, by multiplying the update in (2.2) on the left by $\boldsymbol{z}_t^T$ and on the right by $\boldsymbol{z}_t$ and noting that $\hat{y}_t = \boldsymbol{z}_t^T A_t \boldsymbol{z}_t$, we obtain the following:

$$\bar{y} = \hat{y}_t - \frac{\eta(\bar{y} - y_t)\hat{y}_t^2}{1 + \eta(\bar{y} - y_t)\hat{y}_t}, \text{ and so } \bar{y} = \frac{\eta y_t \hat{y}_t - 1 + \sqrt{(\eta y_t \hat{y}_t - 1)^2 + 4\eta \hat{y}_t^2}}{2\eta \hat{y}_t}. \tag{2.3}$$

We can solve directly for $\bar{y}$ using this formula, and then plug this into the update (2.2). For the case when the input is a quadruple and the loss function is the squared hinge loss, we only perform the update (2.2) if the new constraint is violated.

It is possible to show that the resulting matrix $A_{t+1}$ is positive definite; the proof appears in our longer version. The fact that this update maintains positive definiteness is a key advantage of our method over existing methods; POLA, for example, requires projection to the positive semidefinite cone via an eigendecomposition. The final loss bound in [4] depends on the regularization parameter $\eta_t$ from each iteration and is in turn dependent on the sequence of constraints, an undesirable property for online algorithms. In contrast, by minimizing the function $f_t$ we designate above in (2.1), our algorithm's updates automatically maintain positive definiteness. This means that the regularization parameter $\eta$ need not be changed according to the current constraint, and the resulting bounds (Section 2.2) and empirical performance are notably stronger.

We refer to our algorithm as LogDet Exact Gradient Online (LEGO), and use this name throughout to distinguish it from POLA [13] (which uses a Frobenius regularization) and the Information Theoretic Metric Learning (ITML)-Online algorithm [4] (which uses an approximation to the gradient).

## 2.2 Analysis

We now briefly analyze the regret bounds for our online metric learning algorithm. Due to space issues, we do not present the full proofs; please see the longer version for further details.

To evaluate the online learner's quality, we want to compare the loss of the online algorithm (which has access to one constraint at a time in sequence) to the loss of the best possible offline algorithm (which has access to all constraints at once). Let $\hat{d}_t = d_{A^*}(\boldsymbol{u}_t, \boldsymbol{v}_t)$ be the learned distance between points $\boldsymbol{u}_t$ and $\boldsymbol{v}_t$ with a fixed positive definite matrix $A^*$, and let $L_{A^*} = \sum_t \ell(\hat{d}_t, y_t)$ be the loss suffered over all $t$ time steps. Note that the loss $L_{A^*}$ is with respect to a single matrix $A^*$, whereas $L_A$ (Section 2.1) is with respect to a matrix that is being updated every time step. Let $A^*$ be the optimal offline solution, i.e. it minimizes total loss incurred ($L_{A^*}$). The goal is to demonstrate that the loss of the online algorithm $L_A$ is competitive with the loss of any offline algorithm. To that end, we now show that $L_A \leq c_1 L_{A^*} + c_2$, where $c_1$ and $c_2$ are constants.

In the result below, we assume that the length of the data points is bounded: $\|\boldsymbol{u}\|_2^2 \leq R$ for all $\boldsymbol{u}$. The following key lemma shows that we can bound the loss at each step of the algorithm:

**Lemma 2.1.** *At each step $t$,*

$$\frac{1}{2}\alpha_t(\hat{y}_t - y_t)^2 - \frac{1}{2}\beta_t(d_{A^*}(\boldsymbol{u}_t, \boldsymbol{v}_t) - y_t)^2 \leq D_{ld}(A^*, A_t) - D_{ld}(A^*, A_{t+1}),$$

*where $0 \leq \alpha_t \leq \dfrac{\eta}{1+\eta\left(\frac{R}{2}+\sqrt{\frac{R^2}{4}+\frac{1}{\eta}}\right)^2}$, $\beta_t = \eta$, and $A^*$ is the optimal offline solution.*

*Proof.* See longer version. □

**Theorem 2.2.**

$$L_A \leq \left(1 + \eta\left(\frac{R}{2} + \sqrt{\frac{R^2}{4} + \frac{1}{\eta}}\right)^2\right)L_{A^*} + \left(\frac{1}{\eta} + \left(\frac{R}{2} + \sqrt{\frac{R^2}{4} + \frac{1}{\eta}}\right)^2\right)D_{ld}(A^*, A_0),$$

*where $L_A = \sum_t \ell(\hat{y}_t, y_t)$ is the loss incurred by the series of matrices $A_t$ generated by Equation (2.3), $A_0 \succ 0$ is the initial matrix, and $A^*$ is the optimal offline solution.*
*Proof.* The bound is obtained by summing the loss at each step using Lemma 2.1:

$$\sum_t \left(\frac{1}{2}\alpha_t(\hat{y}_t - y_t)^2 - \frac{1}{2}\beta_t(d_{A^*}(\boldsymbol{u}_t, \boldsymbol{v}_t) - y_t)^2\right) \leq \sum_t \left(D_{ld}(A^*, A_t) - D_{ld}(A^*, A_{t+1})\right).$$

The result follows by plugging in the appropriate $\alpha_t$ and $\beta_t$, and observing that the right-hand side telescopes to $D_{ld}(A^*, A_0) - D_{ld}(A^*, A_{t+1}) \leq D_{ld}(A^*, A_0)$ since $D_{ld}(A^*, A_{t+1}) \geq 0$. □

For the squared hinge loss $\ell(\hat{y}_t, y_t, b_t) = \max(0, b_t(\hat{y}_t - y_t))^2$, the corresponding algorithm has the same bound.

The regularization parameter affects the tradeoff between $L_{A^*}$ and $D_{ld}(A^*, A_0)$: as $\eta$ gets larger, the coefficient of $L_{A^*}$ grows while the coefficient of $D_{ld}(A^*, A_0)$ shrinks. In most scenarios, $R$ is small; for example, in the case when $R = 2$ and $\eta = 1$, then the bound is $L_A \leq (4 + \sqrt{2})L_{A^*} + 2(4 + \sqrt{2})D_{ld}(A^*, A_0)$. Furthermore, in the case when there exists an offline solution with zero error, i.e., $L_{A^*} = 0$, then with a sufficiently large regularization parameter, we know that $L_A \leq 2R^2 D_{ld}(A^*, A_0)$. This bound is analogous to the bound proven in Theorem 1 of the POLA method [13]. Note, however, that our bound is much more favorable to scaling of the optimal solution $A^*$, since the bound of POLA has a $\|A^*\|_F^2$ term while our bound uses $D_{ld}(A^*, A_0)$: if we scale the optimal solution by $c$, then the $D_{ld}(A^*, A_0)$ term will scale by $O(c)$, whereas $\|A^*\|_F^2$ will scale by $O(c^2)$. Similarly, our bound is tighter than that provided by the ITML-Online algorithm since, in the ITML-Online algorithm, the regularization parameter $\eta_t$ for step $t$ is dependent on the input data. An adversary can always provide an input $(\boldsymbol{u}_t, \boldsymbol{v}_t, y_t)$ so that the regularization

parameter has to be decreased arbitrarily; that is, the need to maintain positive defininteness for each update can prevent ITML-Online from making progress towards an optimal metric.

In summary, we have proven a regret bound for the proposed LEGO algorithm, an online metric learning algorithm based on LogDet regularization and gradient descent. Our algorithm automatically enforces positive definiteness every iteration and is simple to implement. The bound is comparable to POLA's bound but is more favorable to scaling, and is stronger than ITML-Online's bound.

# 3 Fast Online Similarity Searches

In many applications, metric learning is used in conjunction with nearest-neighbor searching, and data structures to facilitate such searches are essential. For online metric learning to be practical for large-scale retrieval applications, we must be able to efficiently index the data as updates to the metric are performed. This poses a problem for most fast similarity searching algorithms, since each update to the online algorithm would require a costly update to their data structures.

Our goal is to avoid expensive naive updates, where all database items are re-inserted into the search structure. We employ *locality-sensitive hashing* to enable fast queries; but rather than re-hash all database examples every time an online constraint alters the metric, we show how to incorporate a second level of hashing that determines which hash bits are changing during the metric learning updates. This allows us to avoid costly exhaustive updates to the hash keys, though occasional updating is required after substantial changes to the metric are accumulated.

## 3.1 Background: Locality-Sensitive Hashing

Locality-sensitive hashing (LSH) [6, 1] produces a binary hash key $H(\boldsymbol{u}) = [h_1(\boldsymbol{u})h_2(\boldsymbol{u})...h_b(\boldsymbol{u})]$ for every data point. Each individual bit $h_i(\boldsymbol{u})$ is obtained by applying the locality sensitive hash function $h_i$ to input $\boldsymbol{u}$. To allow sub-linear time approximate similarity search for a similarity function 'sim', a locality-sensitive hash function must satisfy the following property: $Pr[h_i(\boldsymbol{u}) = h_i(\boldsymbol{v})] = \text{sim}(\boldsymbol{u}, \boldsymbol{v})$, where 'sim' returns values between 0 and 1. This means that the more similar examples are, the more likely they are to collide in the hash table.

A LSH function when 'sim' is the inner product was developed in [1], in which a hash bit is the sign of an input's inner product with a random hyperplane. For Mahalanobis distances, the similarity function of interest is $\text{sim}(\boldsymbol{u}, \boldsymbol{v}) = \boldsymbol{u}^T A \boldsymbol{v}$. The hash function in [1] was extended to accommodate a Mahalanobis similarity function in [9]: $A$ can be decomposed as $G^T G$, and the similarity function is then equivalently $\tilde{\boldsymbol{u}}^T \tilde{\boldsymbol{v}}$, where $\tilde{\boldsymbol{u}} = G\boldsymbol{u}$ and $\tilde{\boldsymbol{v}} = G\boldsymbol{v}$. Hence, a valid LSH function for $\boldsymbol{u}^T A \boldsymbol{v}$ is:

$$h_{\boldsymbol{r},A}(\boldsymbol{u}) = \begin{cases} 1, & \text{if } \boldsymbol{r}^T G\boldsymbol{u} \geq 0 \\ 0, & \text{otherwise,} \end{cases} \tag{3.1}$$

where $\boldsymbol{r}$ is the normal to a random hyperplane. To perform sub-linear time nearest neighbor searches, a hash key is produced for all $n$ data points in our database. Given a query, its hash key is formed and then, an appropriate data structure can be used to extract potential nearest neighbors (see [6, 1] for details). Typically, the methods search only $O(n^{1/(1+\epsilon)})$ of the data points, where $\epsilon > 0$, to retrieve the $(1 + \epsilon)$-nearest neighbors with high probability.

## 3.2 Online Hashing Updates

The approach described thus far is not immediately amenable to online updates. We can imagine producing a series of LSH functions $h_{\boldsymbol{r}_1,A}, ..., h_{\boldsymbol{r}_b,A}$, and storing the corresponding hash keys for each data point in our database. However, the hash functions as given in (3.1) are dependent on the Mahalanobis distance; when we update our matrix $A_t$ to $A_{t+1}$, the corresponding hash functions, parameterized by $G_t$, must also change. To update all hash keys in the database would require $O(nd)$ time, which may be prohibitive. In the following we propose a more efficient approach.

Recall the update for $A$: $A_{t+1} = A_t - \frac{\eta(\bar{y}-y_t)A_t \boldsymbol{z}_t \boldsymbol{z}_t^T A_t}{1+\eta(\bar{y}-y_t)\hat{y}_t}$, which we will write as $A_{t+1} = A_t + \beta_t A_t \boldsymbol{z}_t \boldsymbol{z}_t^T A_t$, where $\beta_t = -\eta(\bar{y} - y_t)/(1 + \eta(\bar{y} - y_t)\hat{y}_t)$. Let $G_t^T G_t = A_t$. Then $A_{t+1} = G_t^T(I + \beta_t G_t \boldsymbol{z}_t \boldsymbol{z}_t^T G_t^T)G_t$. The square-root of $I + \beta_t G_t \boldsymbol{z}_t \boldsymbol{z}_t^T G_t^T$ is $I + \alpha_t G_t \boldsymbol{z}_t \boldsymbol{z}_t^T G_t^T$, where $\alpha_t = (\sqrt{1 + \beta_t \boldsymbol{z}_t^T A_t \boldsymbol{z}_t} - 1)/(\boldsymbol{z}_t^T A_t \boldsymbol{z}_t)$. As a result, $G_{t+1} = G_t + \alpha_t G_t \boldsymbol{z}_t \boldsymbol{z}_t^T A_t$. The corresponding update to (3.1) is to find the sign of

$$\boldsymbol{r}^T G_{t+1} \boldsymbol{x} = \boldsymbol{r}^T G_t \boldsymbol{u} + \alpha_t \boldsymbol{r}^T G_t \boldsymbol{z}_t \boldsymbol{z}_t^T A_t \boldsymbol{u}. \tag{3.2}$$

Suppose that the hash functions have been updated in full at some time step $t_1$ in the past. Now at time $t$, we want to determine which hash bits have flipped since $t_1$, or more precisely, which examples' product with some $\boldsymbol{r}^T G_t$ has changed from positive to negative, or vice versa. This amounts to determining all bits such that $\text{sign}(\boldsymbol{r}^T G_{t_1} \boldsymbol{u}) \neq \text{sign}(\boldsymbol{r}^T G_t \boldsymbol{u})$, or equivalently, $(\boldsymbol{r}^T G_{t_1} \boldsymbol{u})(\boldsymbol{r}^T G_t \boldsymbol{u}) \leq 0$. Expanding the update given in (3.2), we can write $\boldsymbol{r}^T G_t \boldsymbol{u}$ as $\boldsymbol{r}^T G_{t_1} \boldsymbol{u} + \sum_{\ell=t_1}^{t-1} \alpha_\ell \boldsymbol{r}^T G_\ell \boldsymbol{z}_\ell \boldsymbol{z}_\ell^T A_\ell \boldsymbol{u}$. Therefore, finding the bits that have changed sign is equivalent to finding all $\boldsymbol{u}$ such that $(\boldsymbol{r}^T G_{t_1} \boldsymbol{u})^2 + (\boldsymbol{r}^T G_{t_1} \boldsymbol{u})\left( \sum_{\ell=t_1}^{t-1} \alpha_\ell \boldsymbol{r}^T G_\ell \boldsymbol{z}_\ell \boldsymbol{z}_\ell^T A_\ell \boldsymbol{u} \right) \leq 0$. We can use a second level of locality-sensitive hashing to *approximately* find all such $\boldsymbol{u}$. Define a vector $\bar{\boldsymbol{u}} = [(\boldsymbol{r}^T G_{t_1} \boldsymbol{u})^2; (\boldsymbol{r}^T G_{t_1} \boldsymbol{u})\boldsymbol{u}]$ and a "query" $\bar{\boldsymbol{q}} = [-1; -\sum_{\ell=t_1}^{t-1} \alpha_\ell \boldsymbol{r}^T A_\ell \boldsymbol{z}_\ell \boldsymbol{z}_\ell^T G_\ell]$. Then the bits that have changed sign can be approximately identified by finding all examples $\bar{\boldsymbol{u}}$ such that $\bar{\boldsymbol{q}}^T \bar{\boldsymbol{u}} \geq 0$. In other words, we look for all $\bar{\boldsymbol{u}}$ that have a large inner product with $\bar{\boldsymbol{q}}$, which translates the problem to a similarity search problem. This may be solved approximately using the locality-sensitive hashing scheme given in [1] for inner product similarity. Note that finding $\bar{\boldsymbol{u}}$ for each $\boldsymbol{r}$ can be computationally expensive, so we search $\bar{\boldsymbol{u}}$ for only a randomly selected subset of the vectors $\boldsymbol{r}$.

In summary, when performing online metric learning updates, instead of updating all the hash keys at every step (which costs $O(nd)$), we delay updating the hash keys and instead determine approximately which bits have changed in the stored entries in the hash table since the last update. When we have a nearest-neighbor query, we can quickly determine which bits have changed, and then use this information to find a query's approximate nearest neighbors using the current metric. Once many of the bits have changed, we perform a full update to our hash functions.

Finally, we note that the above can be extended to the case where computations are done in kernel space. We omit details due to lack of space.

## 4   Experimental Results

In this section we evaluate the proposed algorithm (LEGO) over a variety of data sets, and examine both its online metric learning accuracy as well as the quality of its online similarity search updates. As baselines, we consider the most relevant techniques from the literature: the online metric learners POLA [13] and ITML-Online [4]. We also evaluate a baseline offline metric learner associated with our method. For all metric learners, we gauge improvements relative to the original (non-learned) Euclidean distance, and our classification error is measured with the $k$-nearest neighbor algorithm.

First we consider the same collection of UCI data sets used in [4]. For each data set, we provide the online algorithms with 10,000 randomly-selected constraints, and generate their target distances as in [4]—for same-class pairs, the target distance is set to be equal to the 5th percentile of all distances in the data, while for different-class pairs, the 95th percentile is used. To tune the regularization parameter $\eta$ for POLA and LEGO, we apply a pre-training phase using 1,000 constraints. (This is not required for ITML-Online, which automatically sets the regularization parameter at each iteration to guarantee positive definiteness). The final metric ($A_T$) obtained by each online algorithm is used for testing ($T$ is the total number of time-steps). The left plot of Figure 1 shows the $k$-nn error rates for all five data sets. LEGO outperforms the Euclidean baseline as well as the other online learners, and even approaches the accuracy of the offline method (see [4] for additional comparable offline learning results using [7, 15]). LEGO and ITML-Online have comparable running times. However, our approach has a significant speed advantage over POLA on these data sets: on average, learning with LEGO is 16.6 times faster, most likely due to the extra projection step required by POLA.

Next we evaluate our approach on a handwritten digit classification task, reproducing the experiment used to test POLA in [13]. We use the same settings given in that paper. Using the MNIST data set, we pose a binary classification problem between each pair of digits (45 problems in all). The training and test sets consist of 10,000 examples each. For each problem, 1,000 constraints are chosen and the final metric obtained is used for testing. The center plot of Figure 1 compares the test error between POLA and LEGO. Note that LEGO beats or matches POLA's test error in 33/45 (73.33%) of the classification problems. Based on the additional baselines provided in [13], this indicates that our approach also fares well compared to other offline metric learners on this data set.

We next consider a set of image patches from the Photo Tourism project [14], where user photos from Flickr are used to generate 3-d reconstructions of various tourist landmarks. Forming the reconstructions requires solving for the correspondence between local patches from multiple images of the same scene. We use the publicly available data set that contains about 300,000 total patches

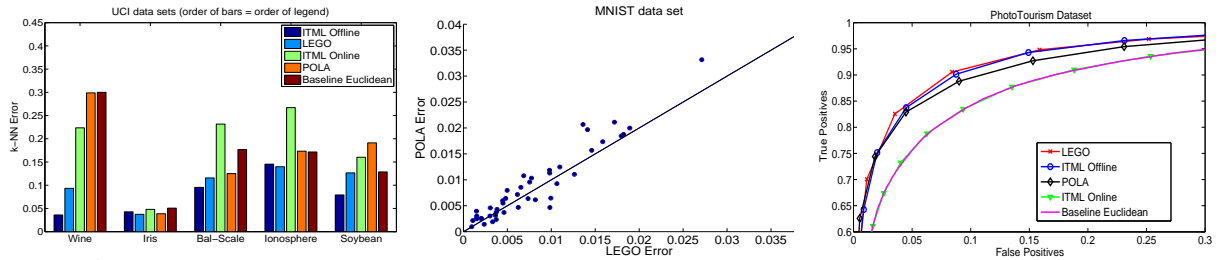

Figure 1: Comparison with existing online metric learning methods. **Left:** On the UCI data sets, our method (LEGO) outperforms both the Euclidean distance baseline as well as existing metric learning methods, and even approaches the accuracy of the offline algorithm. **Center:** Comparison of errors for LEGO and POLA on 45 binary classification problems using the MNIST data; LEGO matches or outperforms POLA on 33 of the 45 total problems. **Right:** On the Photo Tourism data, our online algorithm significantly outperforms the $L_2$ baseline and ITML-Online, does well relative to POLA, and nearly matches the accuracy of the offline method.

from images of three landmarks[1]. Each patch has a dimensionality of 4096, so for efficiency we apply all algorithms in kernel space, and use a linear kernel. The goal is to learn a metric that measures the distance between image patches better than $L_2$, so that patches of the same 3-d scene point will be matched together, and (ideally) others will not. Since the database is large, we can also use it to demonstrate our online hash table updates. Following [8], we add random jitter (scaling, rotations, shifts) to all patches, and generate 50,000 patch constraints (50% matching and 50% non-matching patches) from a mix of the Trevi and Halfdome images. We test with 100,000 patch pairs from the Notre Dame portion of the data set, and measure accuracy with precision and recall.

The right plot of Figure 1 shows that LEGO and POLA are able to learn a distance function that significantly outperforms the baseline squared Euclidean distance. However, LEGO is more accurate than POLA, and again nearly matches the performance of the offline metric learning algorithm. On the other hand, the ITML-Online algorithm does not improve beyond the baseline. We attribute the poor accuracy of ITML-Online to its need to continually adjust the regularization parameter to maintain positive definiteness; in practice, this often leads to significant drops in the regularization parameter, which prevents the method from improving over the Euclidean baseline. In terms of training time, on this data LEGO is 1.42 times faster than POLA (on average over 10 runs).

Finally, we present results using our online metric learning algorithm together with our online hash table updates described in Section 3.2 for the Photo Tourism data. For our first experiment, we provide each method with 50,000 patch constraints, and then search for nearest neighbors for 10,000 test points sampled from the Notre Dame images. Figure 2 (left plot) shows the recall as a function of the number of patches retrieved for four variations: LEGO with a linear scan, LEGO with our LSH updates, the $L_2$ baseline with a linear scan, and $L_2$ with our LSH updates. The results show that the accuracy achieved by our LEGO+LSH algorithm is comparable to the LEGO+linear scan (and similarly, $L_2$+LSH is comparable to $L_2$+linear scan), thus validating the effectiveness of our online hashing scheme. Moreover, LEGO+LSH needs to search only $10\%$ of the database, which translates to an approximate speedup factor of 4.7 over the linear scan for this data set.

Next we show that LEGO+LSH performs accurate and efficient retrievals in the case where constraints and queries are interleaved in any order. Such a scenario is useful in many applications: for example, an image retrieval system such as Flickr continually acquires new image tags from users (which could be mapped to similarity constraints), but must also continually support intermittent user queries. For the Photo Tourism setting, it would be useful in practice to allow new constraints indicating true-match patch pairs to stream in while users continually add photos that should participate in new 3-d reconstructions with the improved match distance functions. To experiment with this scenario, we randomly mix online additions of 50,000 constraints with 10,000 queries, and measure performance by the recall value for 300 retrieved nearest neighbor examples. We recompute the hash-bits for all database examples if we detect changes in more than 10% of the database examples. Figure 2 (right plot) compares the average recall value for various methods after each query. As expected, as more constraints are provided, the LEGO-based accuracies all improve (in contrast to the static $L_2$ baseline, as seen by the straight line in the plot). Our method achieves similar accuracy to both the linear scan method (LEGO Linear) as well as the naive LSH method where the hash table is fully recomputed after every constraint update (LEGO Naive LSH). The curves stack up

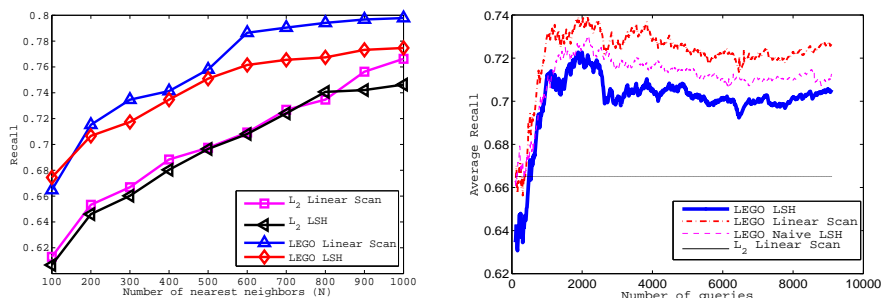

Figure 2: Results with online hashing updates. The left plot shows the recall value for increasing numbers of nearest neighbors retrieved. 'LEGO LSH' denotes LEGO metric learning in conjunction with online searches using our LSH updates, 'LEGO Linear' denotes LEGO learning with linear scan searches. $L_2$ denotes the baseline Euclidean distance. The right plot shows the average recall values for all methods at different time instances as more queries are made and more constraints are added. Our online similarity search updates make it possible to efficiently interleave online learning and querying. See text for details.

appropriately given the levels of approximation: LEGO Linear yields the upper bound in terms of accuracy, LEGO Naive LSH with its exhaustive updates is slightly behind that, followed by our LEGO LSH with its partial and dynamic updates. In reward for this minor accuracy loss, however, our method provides a speedup factor of 3.8 over the naive LSH update scheme. (In this case the naive LSH scheme is actually slower than a linear scan, as updating the hash tables after every update incurs a large overhead cost.) For larger data sets, we can expect even larger speed improvements.

**Conclusions:** We have developed an online metric learning algorithm together with a method to perform online updates to fast similarity search structures, and have demonstrated their applicability and advantages on a variety of data sets. We have proven regret bounds for our online learner that offer improved reliability over state-of-the-art methods in terms of regret bounds, and empirical performance. A disadvantage of our algorithm is that the LSH parameters, e.g. $\epsilon$ and the number of hash-bits, need to be selected manually, and may depend on the final application. For future work, we hope to tune the LSH parameters automatically using a deeper theoretical analysis of our hash key updates in conjunction with the relevant statistics of the online similarity search task at hand.

**Acknowledgments:** This research was supported in part by NSF grant CCF-0431257, NSF-ITR award IIS-0325116, NSF grant IIS-0713142, NSF CAREER award 0747356, Microsoft Research, and the Henry Luce Foundation.

## Footnotes

[1]http://phototour.cs.washington.edu/patches/default.htm

## References

[1] M. Charikar. Similarity Estimation Techniques from Rounding Algorithms. In *STOC*, 2002.

[2] L. Cheng, S. V. N. Vishwanathan, D. Schuurmans, S. Wang, and T. Caelli. Implicit Online Learning with Kernels. In *NIPS*, 2006.

[3] M. Datar, N. Immorlica, P. Indyk, and V. Mirrokni. Locality-Sensitive Hashing Scheme Based on p-Stable Distributions. In *SOCG*, 2004.

[4] J. Davis, B. Kulis, P. Jain, S. Sra, and I. Dhillon. Information-Theoretic Metric Learning. In *ICML*, 2007.

[5] A. Frome, Y. Singer, and J. Malik. Image retrieval and classification using local distance functions. In *NIPS*, 2007.

[6] A. Gionis, P. Indyk, and R. Motwani. Similarity Search in High Dimensions via Hashing. In *VLDB*, 1999.

[7] A. Globerson and S. Roweis. Metric Learning by Collapsing Classes. In *NIPS*, 2005.

[8] G. Hua, M. Brown, and S. Winder. Discriminant embedding for local image descriptors. In *ICCV*, 2007.

[9] P. Jain, B. Kulis, and K. Grauman. Fast Image Search for Learned Metrics. In *CVPR*, 2008.

[10] J. Kivinen and M. K. Warmuth. Exponentiated Gradient Versus Gradient Descent for Linear Predictors. *Inf. Comput.*, 132(1):1–63, 1997.

[11] M. Schultz and T. Joachims. Learning a Distance Metric from Relative Comparisons. In *NIPS*, 2003.

[12] S. Shalev-Shwartz and Y. Singer. Online Learning meets Optimization in the Dual. In *COLT*, 2006.

[13] S. Shalev-Shwartz, Y. Singer, and A. Ng. Online and Batch Learning of Pseudo-metrics. In *ICML*, 2004.

[14] N. Snavely, S. Seitz, and R. Szeliski. Photo Tourism: Exploring Photo Collections in 3D. In *SIGGRAPH Conference Proceedings*, pages 835–846, New York, NY, USA, 2006. ACM Press. ISBN 1-59593-364-6.

[15] K. Weinberger, J. Blitzer, and L. Saul. Distance Metric Learning for Large Margin Nearest Neighbor Classification. In *NIPS*, 2006.

[16] E. Xing, A. Ng, M. Jordan, and S. Russell. Distance Metric Learning, with Application to Clustering with Side-Information. In *NIPS*, 2002.

